# Learning Transformational Invariants from Natural Movies

**Charles F. Cadieu & Bruno A. Olshausen**
Helen Wills Neuroscience Institute
University of California, Berkeley
Berkeley, CA 94720
{cadieu, baolshausen}@berkeley.edu

## Abstract

We describe a hierarchical, probabilistic model that learns to extract complex motion from movies of the natural environment. The model consists of two hidden layers: the first layer produces a sparse representation of the image that is expressed in terms of local amplitude and phase variables. The second layer learns the higher-order structure among the time-varying phase variables. After training on natural movies, the top layer units discover the structure of phase-shifts within the first layer. We show that the top layer units encode *transformational invariants*: they are selective for the speed and direction of a moving pattern, but are invariant to its spatial structure (orientation/spatial-frequency). The diversity of units in both the intermediate and top layers of the model provides a set of testable predictions for representations that might be found in V1 and MT. In addition, the model demonstrates how feedback from higher levels can influence representations at lower levels as a by-product of inference in a graphical model.

## 1  Introduction

A key attribute of visual perception is the ability to extract *invariances* from visual input. In the realm of object recognition, the goal of invariant representation is quite clear: a successful object recognition system must be invariant to image variations resulting from different views of the same object. While spatial invariants are essential for forming a useful representation of the natural environment, there is another, equally important form of visual invariance, namely *transformational invariance*. A transformational invariant refers to the dynamic visual structure that remains the same when the spatial structure changes. For example, the property that a soccer ball moving through the air shares with a football moving through the air is a transformational invariant; it is specific to how the ball moves but invariant to the shape or form of the object. Here we seek to learn such invariants from the statistics of natural movies.

There have been numerous efforts to learn spatial invariants [1, 2, 3] from the statistics of natural images, especially with the goal of producing representations useful for object recognition [4, 5, 6]. However, there have been few attempts to learn transformational invariants from natural sensory data. Previous efforts have either relied on using unnatural, hand-tuned stimuli [7, 8, 9], or unrealistic supervised learning algorithms using only rigid translation of an image [10]. Furthermore, it is unclear to what extent these models have captured the diversity of transformations in natural visual scenes or to what level of abstraction their representations produce transformational invariants.

Previous work learning sparse codes of image sequences has shown that it is possible to recover local, direction-selective components (akin to translating Gabors) [11]. However, this type of model does not capture the abstract property of motion because each unit is bound to a specific orientation, spatial-frequency and location within the image—i.e., it still suffers from the aperture problem.

Here we describe a hierarchical probabilistic generative model that learns transformational invariants from unsupervised exposure to natural movies. A key aspect of the model is the *factorization* of visual information into form and motion, as compared to simply extracting these properties separately. The latter approach characterizes most models of form and motion processing in the visual cortical hierarchy [6, 12], but suffers from the fact that information about these properties is not bound together—i.e., it is not possible to reconstruct an image sequence from a representation in which form and motion have been extracted by separate and independent mechanisms. While reconstruction is not the goal of vision, the ability to interact with the environment is key, and thus binding these properties together is likely to be crucial for properly interacting with the world. In the model we propose here, form and motion are factorized, meaning that extracting one property depends upon the other. It specifies not only how they are extracted, but how they are combined to provide a full description of image content.

We show that when such a model is adapted to natural movies, the top layer units learn to extract transformational invariants. The diversity of units in both the intermediate layer and top layer provides a set of testable predictions for representations that might be found in V1 and MT. The model also demonstrates how feedback from higher levels can influence representations at lower levels as a by-product of inference in a graphical model.

## 2 Hierarchical Model

In this section we introduce our hierarchical generative model of time-varying images. The model consists of an input layer and two hidden layers as shown in Figure 1. The input layer represents the time-varying image pixel intensities. The first hidden layer is a sparse coding model utilizing complex basis functions, and shares many properties with subspace-ICA [13] and the standard energy model of complex cells [14]. The second hidden layer models the dynamics of the complex basis function phase variables.

### 2.1 Sparse coding with complex basis functions

In previous work it has been shown that many of the observed response properties of neurons in V1 may be accounted for in terms of a sparse coding model of images [15, 16]:

$$I_{(x,t)} = \sum_i u_i(t)\, A_i(x) + n_{(x,t)} \tag{1}$$

where $I_{(x,t)}$ is the image intensity as a function of space ($x \in \mathcal{R}^2$) and time, $A_i(x)$ is a spatial basis function with coefficient $u_i$, and the term, $n_{(x,t)}$ corresponds to Gaussian noise with variance $\sigma_n^2$ that is small compared to the image variance. The sparse coding model imposes a kurtotic, independent prior over the coefficients, and when adapted to natural image patches the $A_i(x)$ converge to a set of localized, oriented, multiscale functions similar to a Gabor wavelet decomposition of images.

We propose here a generalization of the sparse coding model to complex variables that is primarily motivated from two observations of natural image statistics. The first observation is that although the prior is factorial, the actual joint distribution of coefficients, even after learning, exhibits strong statistical dependencies. These are most clearly seen as circularly symmetric, yet kurtotic distributions among pairs of coefficients corresponding to neighboring basis functions, as first described by Zetzsche [17]. Such a circularly symmetric distribution strongly suggests that these pairs of coefficients are better described in polar coordinates rather than Cartesian coordinates—i.e., in terms of *amplitude* and *phase*. The second observation comes from considering the dynamics of coefficients through time. As pointed out by Hyvarinen [3], the temporal evolution of a coefficient in response to a movie, $u_i(t)$, can be well described in terms of the product of a smooth amplitude envelope multiplied by a quickly changing variable. A similar result from Kording [1] indicates that temporal continuity in amplitude provides a strong cue for learning local invariances. These results are closely related to the trace learning rule of Foldiak [18] and slow feature analysis [19].

With these observations in mind, we have modified the sparse coding model by utilizing a complex basis function model as follows:

$$I_{(x,t)} = \sum_i \Re\{z_i^*(t)\, A_i(x)\} + n_{(x,t)} \tag{2}$$

where the basis functions now have real and imaginary parts, $A_i(x) = A_i^{\mathcal{R}}(x) + jA_i^{\mathcal{I}}(x)$, and the coefficients are also complex, with $z_i(t) = a_i(t)e^{j\phi_i(t)}$. ($*$ indicates the complex conjugate and the notation $\Re\{.\}$ denotes taking the 'real part' of the argument.) The resulting generative model can also be written as:

$$I_{(x,t)} = \sum_i a_{i(t)} \left[ \cos\phi_{i(t)}\, A_i^{\mathcal{R}}(x) + \sin\phi_{i(t)}\, A_i^{\mathcal{I}}(x) \right] + n_{(x,t)} \tag{3}$$

Thus, each pair of basis functions $A_i^{\mathcal{R}}, A_i^{\mathcal{I}}$ forms a 2-dimensional subspace and is controlled by an amplitude $a_i$ and phase $\phi_i$ that determine the position within each subspace. Note that the basis functions are only functions of space. Therefore, the temporal dynamics within image sequences will be expressed in the temporal dynamics of the amplitude and phase.

The prior over the complex coefficients, $z$, is designed so as to enforce circularly symmetric distributions and smooth amplitude dynamics as observed from time-varying natural images:

$$P(a_{i(t)}|a_{i(t-1)}) \propto e^{-Sp_a(a_{i(t)}) - Sl_a(a_{i(t)}, a_{i(t-1)})} \tag{4}$$

The first term in the exponential imposes a sparse prior on the coefficient amplitudes. Here we use $Sp(a_{i(t)}) = \lambda a_{i(t)}$ (we have found other kurtotic priors to yield similar results). Since there is no prior over the phases, this will result in circularly symmetric kurtotic distributions over each subspace. The second term in the exponential imposes temporal stability on the time rate of change of the amplitudes and is given by: $Sl_a(a_{i(t)}, a_{i(t-1)}) = (a_{i(t)} - a_{i(t-1)})^2$.

For a sequence of images the resulting negative log-posterior for the first hidden layer becomes:

$$E_1 = \sum_t \sum_x \tfrac{1}{\sigma_N^2} \left[ I_{(x,t)} - \sum_i \Re\{z_i^*(t)\, A_i(x)\} \right]^2 + \sum_{i,t} Sp(a_{i(t)}) + \sum_{i,t} Sl(a_{i(t)}, a_{i(t-1)}) \tag{5}$$

While this model by no means captures the full joint distribution of coefficients, it does at least capture the circular symmetric dependencies among pairs of coefficients, which allows for the explicit representation of amplitude and phase. As we shall see, this representation serves as a staging ground for learning higher-order dependencies over space and time.

## 2.2 Phase Transformations

Given the decomposition into amplitude and phase variables, we now have a non-linear representation of image content that enables us to learn its structure in another linear generative model. In particular, the dynamics of objects moving in continuous trajectories through the world over short epochs will be encoded in the population activity of the phase variables $\phi_i$. Furthermore, because we have encoded these trajectories with an angular variable, many transformations in the image domain that would otherwise be nonlinear in the coefficients $u_i$ will now be linearized. This linear relationship allows us to model the time-rate of change of the phase variables with a simple linear generative model.

We thus model the first-order time derivative of the phase variables as follows:

$$\dot{\phi}_i(t) = \sum_k D_{ik}\, w_k(t) + \nu_i(t) \tag{6}$$

where $\dot{\phi}_i = \phi_i(t) - \phi_i(t-1)$, and $D$ is the basis function matrix specifying how the high-level variables $w_k$ influence the phase shifts $\dot{\phi}_i$. The additive noise term, $\nu_i$, represents uncertainty or noise in the estimate of the phase time-rate of change. As before, we impose a sparse, independent distribution on the coefficients $w_k$, in this case with a sparse cost function given as:

$$S_w(w_k(t)) = \beta \log\left(1 + \left(\frac{w_k(t)}{\sigma}\right)^2\right) \tag{7}$$

The uncertainty over the phase shifts is given by a von Mises distribution: $p(\nu_i) \propto \exp(\kappa \cos(\nu_i))$. Thus, the log-posterior over the second layer units is given by

$$E_2 = -\sum_t \sum_{i \in \{a_i(t) > 0\}} \kappa \cos(\dot{\phi}_i - [Dw(t)]_i) + \sum_k S_w(w_k(t)) \tag{8}$$

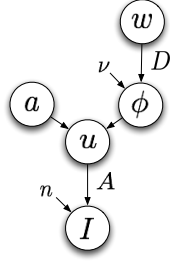

Figure 1: Graph of the hierarchical model showing the relationship among hidden variables.

Because the angle of a variable with 0 amplitude is undefined, we exclude angles where the corresponding amplitude is 0 from our cost function.

Note that in the first layer we did not introduce any prior on the phase variables. With our second hidden layer, $E_2$ can be viewed as a log-prior on the time rate of change of the phase variables: $\dot{\phi}_{i(t)}$. For example, when $[Dw(t)]_i = 0$, the prior on $\dot{\phi}_{i(t)}$ is peaked around 0, or no change in phase. Activating the $w$ variables moves the prior away from $\dot{\phi}_{i(t)} = 0$, encouraging certain patterns of phase shifting that will in turn produce patterns of motion in the image domain.

The structure of the complete graphical model is shown in Figure 1.

### 2.3 Learning and inference

A variational learning algorithm is used to adapt the basis functions in both layers. First we infer the maximum *a posteriori* estimate of the variables $a$, $\phi$, and $w$ for the current values of the basis functions. Given the map estimate of these variables we then perform a gradient update on the basis functions. The two steps are iterated until convergence.

To infer coefficients in both the first and second hidden layers we perform gradient descent with respect to the coefficients of the total cost function ($E_1 + E_2$). The resulting dynamics for the amplitudes and phases in the first layer are given by

$$\Delta a_{i(t)} \quad \propto \quad \Re\{b_{i(t)}\} - Sp'(a_{i(t)}) - Sl'(a_{i(t)}, a_{i(t-1)}) \tag{9}$$

$$\Delta \phi_{i(t)} \quad \propto \quad \Im\{b_{i(t)}\} a_{i(t)} - \kappa \sin(\dot{\phi}_{i(t)} - [Dw(t)]_i) + \kappa \sin(\dot{\phi}_{i(t+1)} - [Dw(t+1)]_i) \tag{10}$$

with $b_{i(t)} = \frac{1}{\sigma_N^2} e^{-j\phi_{i(t)}} \sum_x A_{i(x)} \left[ I_{(x,t)} - \sum_i \Re\{z_i^*(t) A_{i(x)}\} \right]$. ($\Im\{.\}$ denotes the imaginary part.)

The dynamics for the second layer coefficients $w_k$ are given by

$$\Delta w_{k(t)} \propto \sum_{i \in \{a_{i(t)} > 0\}} \kappa \sin(\dot{\phi}_i - [Dw(t)]_i) D_{ik} + S_w'(w_{k(t)}) \tag{11}$$

Note that the two hidden layers are coupled, since the inference of $w$ depends on $\phi$, and the inference of $\phi$ in turn depends on $w$, in addition to $I$ and $a$. Thus, the phases are computed from a combination of bottom-up ($I$), horizontal ($a$) and top-down ($w$) influences.

The learning rule for the first layer basis functions is given by the gradient of $E_1$ with respect to $A_{i(x)}$, using the values of the complex coefficients inferred in eqs. 9 and 10 above:

$$\Delta A_{i(x)} \propto \frac{1}{\sigma_N^2} \sum_t \left[ I_{(x,t)} - \sum_i \Re\{z_i^*(t) A_{i(x)}\} \right] z_{i(t)} \tag{12}$$

The learning rule for the second layer basis functions is given by the gradient of $E_2$ with respect to $D$, using the values of $\phi$ and $w$ inferred above:

$$\Delta D_{ik} = \kappa \sum_{t \in a_{i(t)} > 0} \sin(\dot{\phi}_i - [Dw(t)]_i) w_{k(t)} \tag{13}$$

After each gradient update the basis functions are normalized to have unit length.

## 3 Results

### 3.1 Simulation procedures

The model was trained on natural image sequences obtained from Hans van Hateren's repository at `http://hlab.phys.rug.nl/vidlib/`. The movies were spatially lowpass filtered and whitened as described previously [15]. Note that no whitening in time was performed since the temporal structure will be learned by the hierarchical model. The movies consisted of footage of animals in grasslands along rivers and streams. They contain a variety of motions due to the movements of animals in the scene, camera motion, tracking (which introduces background motion), and motion borders due to occlusion.

We trained the first layer of the model on 20x20 pixel image patches, using 400 complex basis functions $A_i$ in the first hidden layer initialized to random values. During this initial phase of learning only the terms in $E_1$ are used to infer the $a_i$ and $\phi_i$. Once the first layer reaches convergence, we begin training the second layer, using 100 bases, $D_i$, initialized to random values. The second layer bases are initially trained on the MAP estimates of the first layer $\dot{\phi}_i$ inferred using $E_1$ only. After the second layer begins to converge we infer coefficients in both the first layer and the second layer simultaneously using all terms in $E_1 + E_2$ (we observed that this improved convergence in the second layer). We then continued learning in both layers until convergence. The bootstrapping of the second layer was used to speed convergence and we did not observe much change in the first layer basis functions after the initial convergence. We have run the algorithm multiple times and have observed qualitatively similar results on each run. Here we describe the results of one run.

### 3.2 Learned complex basis functions

After learning, the first layer complex basis functions converge to a set of localized, oriented, and bandpass functions with real and imaginary parts roughly in quadrature. The population of filters as a whole tile the joint spaces of orientation, position, and center spatial frequency. Not surprisingly, this result shares similarities to previous results described in [1] and [3]. Figure 2(a) shows the real part, imaginary part, amplitude, and angle of two representative basis functions as a function of space. Examining the amplitude of the basis function we see that it is localized and has a roughly Gaussian envelope. The angle as a function of space reveals a smooth ramping of the phase in the direction perpendicular to the basis functions' orientation.

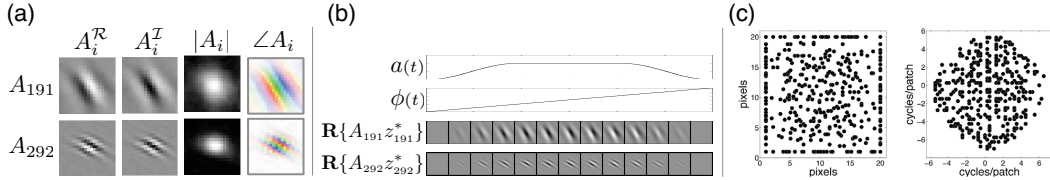

Figure 2: Learned Complex Basis Functions (for panel (b) see the animation in `movie_TransInv_Figure2.mov`).

A useful way of visualizing what a generative model has learned is to generate images while varying the coefficients. Figure 2(b) displays the resulting image sequences produced by two representative basis functions as the amplitude and phase follow the indicated time courses. The amplitude has the effect of controlling the presence of the feature within the image and the phase is related to the position of the edge within the image. Importantly for our hierarchical model, the time derivative, or slope of the phase through time is directly related to the movement of the edge through time.

Figure 2(c) shows how the population of complex basis functions tiles the space of position (left) and spatial-frequency (right). Each dot represents a different basis function according to its maximum amplitude in the space domain, or its maximum amplitude in the frequency domain computed via the 2D Fourier transform of each complex pair (which produces a single peak in the spatial-frequency plane). The basis functions uniformly tile both domains. This visualization will be useful for understanding what the phase shifting components $D$ in the second layer have learned.

### 3.3 Learned phase-shift components

Figure 3 shows a random sampling of 16 of the learned phase-shift components, $D_i$, visualized in both the space domain and frequency domain depictions of the first-layer units. The strength of connection for each component is denoted by hue (red +, blue -, gray 0). Some have a global influence over all spatial positions within the 20x20 input array (e.g., row 1, column 1), while others have influence only over a local region (e.g., row 1, column 6). Those with a linear ramp in the Fourier domain correspond to rigid translation, since the higher spatial-frequencies will spin their phases at proportionally higher rates (and negative spatial-frequencies will spin in the opposite direction). Some functions we believe arise from aliased temporal structure in the movies (row 1, column 5), and others are unknown (row 2, column 4). We are actively seeking methods to quantify these classes of learned phase-shift components.

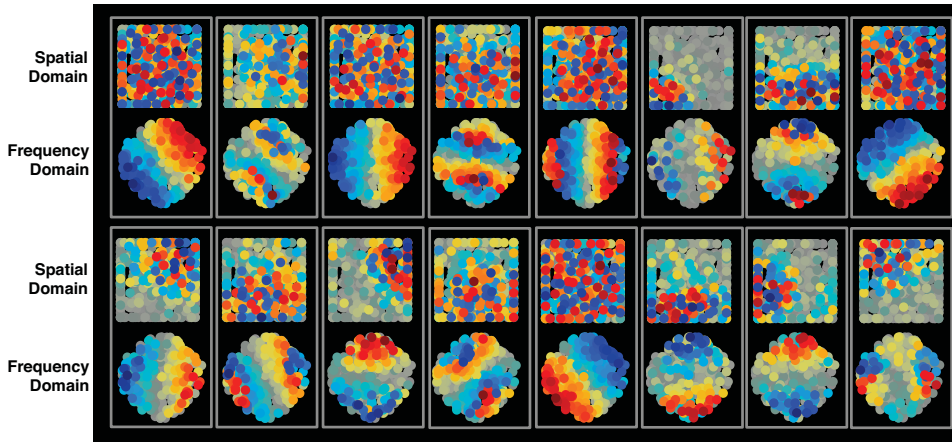

Figure 3: Learned phase shifting components.

The phase shift components generate movements within the image that are invariant to aspects of the spatial structure such as orientation and spatial-frequency. We demonstrate this in Figure 4 by showing the generated transforms for 4 representative phase-shift components. The illustrated transformation components produce: (a) global translation, (b) local translation, (c) horizontal dilation and contraction, and (d) local warping. See the caption of Figure 4 for a more detailed description of the generated motions. We encourage the reader to view the accompanying videos.

## 4 Discussion and conclusions

The computational vision community has spent considerable effort on developing motion models. Of particular relevance to our work is the Motion-Energy model [14], which signals motion via the *amplitudes* of quadrature pair filter outputs, similar to the responses of complex neurons in V1. Simoncelli & Heeger have shown how it is possible to extract motion by pooling over a population of such units lying within a common plane in the 3D Fourier domain [12]. It has not been shown how the representations in these models could be learned from natural images. Furthermore, it is unclear how more complicated transformations, other than local translations, would be represented by such a model, or indeed how the entire joint space of position, direction and speed should be tiled to provide a complete description of time-varying images. Our model addresses each of these problems: it learns from the statistics of natural movies how to best tile the joint domain of position and motion, and it captures complex motion beyond uniform translation.

Central to our model is the representation of *phase*. The use of phase information for computing motion is not new, and was used by Fleet and Jepson [20] to compute optic flow. In addition, as shown in Eero Simoncelli's Thesis, one can establish a formal equivalence between phase-based methods and motion energy models. Here we argue that phase provides a convenient representation as it linearizes trajectories in coefficient space and thus allows one to capture the higher-order structure via a simple linear generative model. Whether or how phase is represented in V1 is not known,

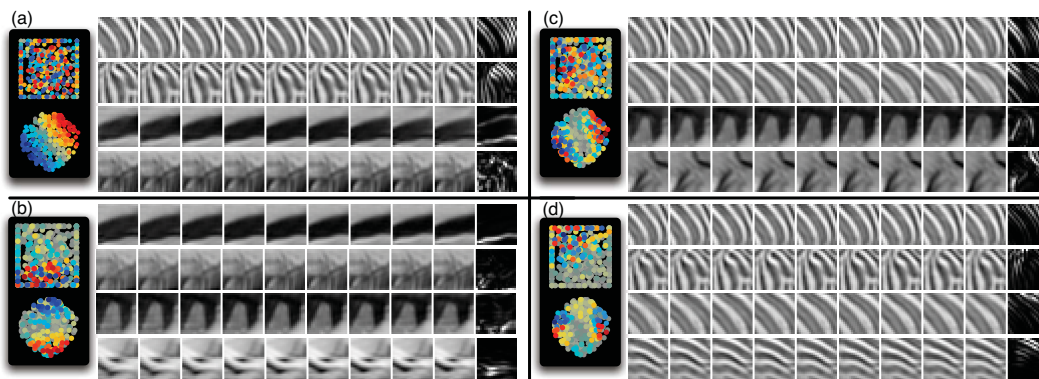

Figure 4: Visualization of learned transformational invariants (best viewed as animations in `movie_TransInv_Figure4x.mov`, x=a,b,c,d). Each phase-shift component produces a pattern of motion that is invariant to the spatial structure contained within the image. Each panel displays the induced image transformations for a different basis function, $D_i$. Induced motions are shown for four different image patches with the original static patch displayed in the center position. Induced motions are produced by turning on the respective coefficient $w_i$ positively (patches to the left of center) and negatively (patches to the right of center). The final image in each sequence shows the pixel-wise variance of the transformation (white values indicate where image pixels are changing through time, which may be difficult to discern in this static presentation). The example in (a) produces global motion in the direction of 45 $\deg$. The strongly oriented structure within the first two patches clearly moves along the axis of motion. Patches with more complicated spatial structure (4th patch) also show similar motion. The next example (b) produces local vertical motion in the lower portion of the image patch only. Note that in the first patch the strong edge in the lower portion of the patch moves while the edge in the upper portion remains fixed. Again, this component produces similar transformations irrespective of the spatial structure contained in the image. The example in (c) produces horizontal motion in the left part of the image in the opposite direction of horizontal motion in the right half (the two halves of the image either converge or diverge). Note that the oriented structure in the first two patches becomes more closely spaced in the leftmost patch and is more widely spaced in the right most image. This is seen clearly in the third image as the spacing between the vertical structure is most narrow in the leftmost image and widest in the rightmost image. The example in (d) produces warping in the upper part of the visual field. This example does not lend itself to a simple description, but appears to produce a local rotation of the image patch.

but it may be worth looking for units that have response properties similar to those of the 'phase units' in our model.

Our model also has implications for other aspects of visual processing and cortical architecture. Under our model we may reinterpret the hypothesized split between the dorsal and ventral visual streams. Instead of independent processing streams focused on form perception and motion perception, the two streams may represent complementary aspects of visual information: *spatial invariants* and *transformational invariants*. Indeed, the pattern-invariant direction tuning of neurons in MT is strikingly similar to that found in our model [21]. Importantly though, in our model information about form and motion is bound together since it is computed by a process of factorization rather than by independent mechanisms in separate streams.

Our model also illustrates a functional role for feedback between higher visual areas and primary visual cortex, not unlike the proposed inference pathways suggested by Lee and Mumford [22]. The first layer units are responsive to visual information in a narrow spatial window and narrow spatial frequency band. However, the top layer units receive input from a diverse population of first layer units and can thus disambiguate local information by providing a bias to the time rate of change of the phase variables. Because the second layer weights $D$ are adapted to the statistics of natural movies, these biases will be consistent with the statistical distribution of motion occurring in the

natural environment. This method can thus deal with artifacts such as noise or temporal aliasing and can be used to disambiguate local motions confounded by the aperture problem.

Our model could be extended in a number of ways. Most obviously, the graphical model in Figure 1 begs the question of what would be gained by modeling the joint distribution over the amplitudes, $a_i$, in addition to the phases. To some degree, this line of approach has already been pursued by Karklin & Lewicki [2], and they have shown that the high level units in this case learn spatial invariants within the image. We are thus eager to combine both of these models into a unified model of higher-order form and motion in images.

## References

[1] W. Einhauser, C. Kayser, P. Konig, and K.P. Kording. Learning the invariance properties of complex cells from their responses to natural stimuli. *European Journal of Neuroscience*, 15(3):475–486, 2002.

[2] Y. Karklin and M.S. Lewicki. A hierarchical bayesian model for learning nonlinear statistical regularities in nonstationary natural signals. *Neural Computation*, 17(2):397–423, 2005.

[3] A. Hyvärinen, J. Hurri, and J. Väyrynen. Bubbles: a unifying framework for low-level statistical properties of natural image sequences. *Journal of the Optical Society of America A*, 20(7):1237–1252, 2003.

[4] G. Wallis and E.T. Rolls. Invariant face and object recognition in the visual system. *Progress in Neurobiology*, 51(2):167–194, 1997.

[5] Y. LeCun, F.J. Huang, and L. Bottou. Learning methods for generic object recognition with invariance to pose and lighting. *Computer Vision and Pattern Recognition*, 2004.

[6] T. Serre, L. Wolf, S. Bileschi, M. Riesenhuber, and T. Poggio. Robust object recognition with cortex-like mechanisms. *IEEE Transactions on Pattern Analysis and Machine Intelligence*, pages 411–426, 2007.

[7] SJ Nowlan and T.J. Sejnowski. A selection model for motion processing in area MT of primates. *Journal of Neuroscience*, 15(2):1195–1214, 1995.

[8] K. Zhang, M. I. Sereno, and M. E. Sereno. Emergence of position-independent detectors of sense of rotation and dilation with Hebbian learning: An analysis. *Neural Computation*, 5(4):597–612, 1993.

[9] E.T. Rolls and S.M. Stringer. Invariant global motion recognition in the dorsal visual system: A unifying theory. *Neural Computation*, 19(1):139–169, 2007.

[10] D.B. Grimes and R.P.N. Rao. Bilinear sparse coding for invariant vision. *Neural Computation*, 17(1):47–73, 2005.

[11] B.A. Olshausen. *Probabilistic Models of Perception and Brain Function*, chapter Sparse codes and spikes, pages 257–272. MIT Press, 2002.

[12] E.P. Simoncelli and D.J. Heeger. A model of neuronal responses in visual area MT. *Vision Research*, 38(5):743–761, 1998.

[13] A. Hyvarinen and P. Hoyer. Emergence of phase-and shift-invariant features by decomposition of natural images into independent feature subspaces. *Neural Computation*, 12(7):1705–1720, 2000.

[14] E.H. Adelson and J.R. Bergen. Spatiotemporal energy models for the perception of motion. *Journal of the Optical Society of America, A*, 2(2):284–299, 1985.

[15] B.A. Olshausen and D.J. Field. Sparse coding with an overcomplete basis set: A strategy employed by v1? *Vision Research*, 37:3311–3325, 1997.

[16] A.J. Bell and T. Sejnowski. The independent components of natural images are edge filters. *Vision Research*, 37:3327–3338, 1997.

[17] C. Zetzsche, G. Krieger, and B. Wegmann. The atoms of vision: Cartesian or polar? *Journal of the Optical Society of America A*, 16(7):1554–1565, 1999.

[18] P. Foldiak. Learning invariance from transformation sequences. *Neural Computation*, 3(2):194–200, 1991.

[19] L. Wiskott and T.J. Sejnowski. Slow feature analysis: Unsupervised learning of invariances. *Neural Computation*, 14(4):715–770, 2002.

[20] D.J. Fleet and A.D. Jepson. Computation of component image velocity from local phase information. *International Journal of Computer Vision*, 5:77–104, 1990.

[21] J.A. Movshon, E.H. Adelson, M.S. Gizzi, and W.T. Newsome. The analysis of moving visual patterns. *Pattern Recognition Mechanisms*, 54:117–151, 1985.

[22] T.S. Lee and D. Mumford. Hierarchical bayesian inference in the visual cortex. *Journal of the Optical Society of America A*, 20(7):1434–1448, 2003.

